# A Non-linear Information Maximisation Algorithm that Performs Blind Separation.

Anthony J. Bell
tony@salk.edu

Terrence J. Sejnowski
terry@salk.edu

Computational Neurobiology Laboratory
The Salk Institute
10010 N. Torrey Pines Road
La Jolla, California 92037-1099

and

Department of Biology
University of California at San Diego
La Jolla CA 92093

## Abstract

A new learning algorithm is derived which performs online stochastic gradient ascent in the mutual information between outputs and inputs of a network. In the absence of *a priori* knowledge about the 'signal' and 'noise' components of the input, propagation of information depends on calibrating network non-linearities to the detailed higher-order moments of the input density functions. By incidentally minimising mutual information between outputs, as well as maximising their individual entropies, the network 'factorises' the input into independent components. As an example application, we have achieved near-perfect separation of ten digitally mixed speech signals. Our simulations lead us to believe that our network performs better at blind separation than the Herault-Jutten network, reflecting the fact that it is derived rigorously from the mutual information objective.

# 1  Introduction

Unsupervised learning algorithms based on information theoretic principles have tended to focus on linear decorrelation (Barlow & Földiák 1989) or maximisation of signal-to-noise ratios assuming Gaussian sources (Linsker 1992). With the exception of (Becker 1992), there has been little attempt to use non-linearity in networks to achieve something a linear network could not.

Non-linear networks, however, are capable of computing more general statistics than those second-order ones involved in decorrelation, and as a consequence they are capable of dealing with signals (and noises) which have detailed higher-order structure. The success of the 'H-J' networks at blind separation (Jutten & Herault 1991) suggests that it should be possible to separate statistically independent components, by using learning rules which make use of moments of all orders.

This paper takes a principled approach to this problem, by starting with the question of how to maximise the information passed on in non-linear feed-forward network. Starting with an analysis of a single unit, the approach is extended to a network mapping $N$ inputs to $N$ outputs. In the process, it will be shown that, under certain fairly weak conditions, the $N \rightarrow N$ network forms a minimally redundant encoding of the inputs, and that it therefore performs Independent Component Analysis (ICA).

# 2  Information maximisation

The information that output $Y$ contains about input $X$ is defined as:

$$I(Y, X) = H(Y) - H(Y|X) \tag{1}$$

where $H(Y)$ is the entropy (information) in the output, while $H(Y|X)$ is whatever information the output has which didn't come from the input. In the case that we have no noise (or rather, we don't know what is noise and what is signal in the input), the mapping between $X$ and $Y$ is deterministic and $H(Y|X)$ has its lowest possible value of $-\infty$. Despite this, we may still differentiate eq.1 as follows (see [5]):

$$\frac{\partial}{\partial w} I(Y, X) = \frac{\partial}{\partial w} H(Y) \tag{2}$$

Thus in the noiseless case, the mutual information can be maximised by maximising the entropy alone.

## 2.1  One input, one output.

Consider an input variable, $x$, passed through a transforming function, $g(x)$, to produce an output variable, $y$, as in Fig.2.1(a). In the case that $g(x)$ is monotonically increasing or decreasing (ie: has a unique inverse), the probability density function (pdf) of the output $f_y(y)$ can be written as a function of the pdf of the input $f_x(x)$, (Papoulis, eq. 5-5):

$$f_y(y) = \frac{f_x(x)}{\partial y / \partial x} \tag{3}$$

The entropy of the output, $H(y)$, is given by:

$$H(y) = -E\left[\ln f_y(y)\right] = -\int_{-\infty}^{\infty} f_y(y) \ln f_y(y) dy \qquad (4)$$

where $E[.]$ denotes expected value. Substituting eq.3 into eq.4 gives

$$H(y) = E\left[\ln \frac{\partial y}{\partial x}\right] - E\left[\ln f_x(x)\right] \qquad (5)$$

The second term on the right may be considered to be unaffected by alterations in a parameter, $w$, determining $g(x)$. Therefore in order to maximise the entropy of $y$ by changing $w$, we need only concentrate on maximising the first term, which is the average log of how the input affects the output. This can be done by considering the 'training set' of $x$'s to approximate the density $f_x(x)$, and deriving an 'online', stochastic gradient descent learning rule:

$$\Delta w \propto \frac{\partial H}{\partial w} = \frac{\partial}{\partial w}\left(\ln \frac{\partial y}{\partial x}\right) = \left(\frac{\partial y}{\partial x}\right)^{-1} \frac{\partial}{\partial w}\left(\frac{\partial y}{\partial x}\right) \qquad (6)$$

In the case of the logistic transfer function $y = (1 + e^{-u})^{-1}$, $u = wx + w_0$ in which the input is multiplied by a weight $w$ and added to a bias-weight $w_0$, the terms above evaluate as:

$$\frac{\partial y}{\partial x} = wy(1 - y) \qquad (7)$$

$$\frac{\partial}{\partial w}\left(\frac{\partial y}{\partial x}\right) = y(1 - y)(1 + wx(1 - 2y)) \qquad (8)$$

Dividing eq.8 by eq.7 gives the learning rule for the logistic function, as calculated from the general rule of eq.6:

$$\Delta w \propto \frac{1}{w} + x(1 - 2y) \qquad (9)$$

Similar reasoning leads to the rule for the bias-weight:

$$\Delta w_0 \propto 1 - 2y \qquad (10)$$

The effect of these two rules can be seen in Fig. 1a. For example, if the input pdf $f_x(x)$ was gaussian, then the $\Delta w_0$-rule would centre the steepest part of the sigmoid curve on the peak of $f_x(x)$, matching input density to output slope, in a manner suggested intuitively by eq.3. The $\Delta w$-rule would then scale the slope of the sigmoid curve to match the variance of $f_x(x)$. For example, narrow pdfs would lead to sharply-sloping sigmoids. The $\Delta w$-rule is basically *anti-Hebbian*[1], with an *anti-decay* term. The anti-Hebbian term keeps $y$ away from one uninformative situation: that of $y$ being saturated to 0 or 1. But anti-Hebbian rules alone make weights go to zero, so the anti-decay term $(1/w)$ keeps $y$ away from the other uninformative situation: when $w$ is so small that $y$ stays around 0.5. The effect of these two balanced forces is to produce an output pdf $f_y(y)$ which is close to the flat unit distribution, which is the maximum entropy distribution for a variable

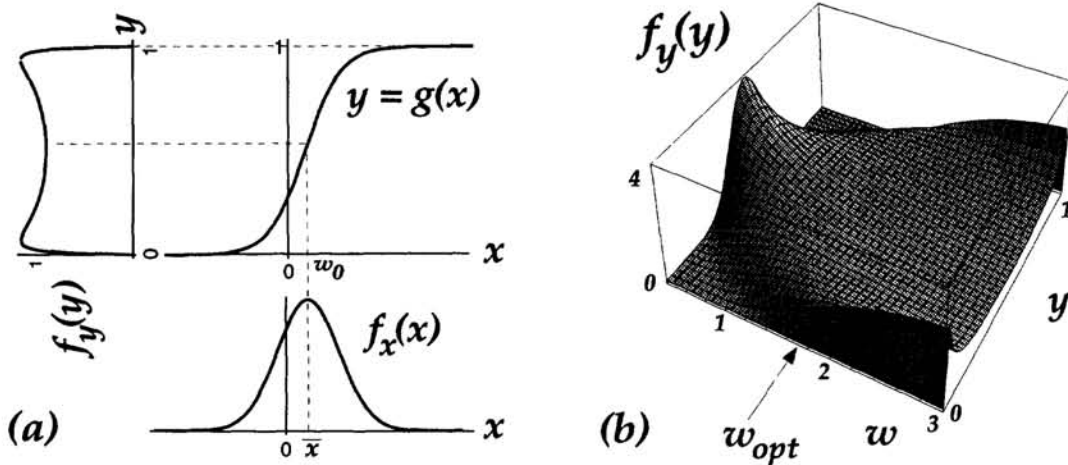

Figure 1: (a) Optimal information flow in sigmoidal neurons (Schraudolph et al 1992). Input $x$ having density function $f_x(x)$, in this case a gaussian, is passed through a non-linear function $g(x)$. The information in the resulting density, $f_y(y)$ depends on matching the mean and variance of $x$ to the threshold and slope of $g(x)$. In (b) $f_y(y)$ is plotted for different values of the weight $w$. The optimal weight, $w_{opt}$ transmits most information.

bounded between 0 and 1. Fig. 1b illustrates a family of these distributions, with the highest entropy one occuring at $w_{opt}$.

A rule which maximises information for one input and one output may be suggestive for structures such as synapses and photoreceptors which must position the gain of their non-linearity at a level appropriate to the average value and size of the input fluctuations. However, to see the advantages of this approach in artificial neural networks, we now analyse the case of multi-dimensional inputs and outputs.

## 2.2   N inputs, N outputs.

Consider a network with an input vector $\mathbf{x}$, a weight matrix $\mathbf{W}$ and a monotonically transformed output vector $\mathbf{y} = g(\mathbf{W}\mathbf{x} + \mathbf{w}_0)$. Analogously to eq.3, the multivariate probability density function of $\mathbf{y}$ can be written (Papoulis, eq. 6-63):

$$f_\mathbf{y}(\mathbf{y}) = \frac{f_\mathbf{x}(\mathbf{x})}{|J|} \qquad (11)$$

where $|J|$ is the absolute value of the Jacobian of the transformation. The Jacobian is the determinant of the matrix of partial derivatives:

$$J = \det \begin{bmatrix} \frac{\partial y_1}{\partial x_1} & \cdots & \frac{\partial y_1}{\partial x_n} \\ \vdots & & \vdots \\ \frac{\partial y_n}{\partial x_1} & \cdots & \frac{\partial y_n}{\partial x_n} \end{bmatrix} \qquad (12)$$

The derivation proceeds as in the previous section except instead of maximising $\ln(\partial y/\partial x)$, now we maximise $\ln|J|$. For sigmoidal units, $\mathbf{y} = (1 + e^{-\mathbf{u}})^{-1}, \mathbf{u} =$

$\mathbf{W}\mathbf{x} + \mathbf{w}_0$, the resulting learning rules are familiar in form:

$$\Delta \mathbf{W} \quad \propto \quad \left[\mathbf{W}^T\right]^{-1} + \mathbf{x}(\mathbf{1} - 2\mathbf{y})^T \tag{13}$$

$$\Delta \mathbf{w}_0 \quad \propto \quad \mathbf{1} - 2\mathbf{y} \tag{14}$$

except that now $\mathbf{x}$, $\mathbf{y}$, $\mathbf{w}_0$ and $\mathbf{1}$ are vectors ($\mathbf{1}$ is a vector of ones), $\mathbf{W}$ is a matrix, and the anti-Hebbian term has become an outer product. The anti-decay term has generalised to the inverse of the transpose of the weight matrix. For an individual weight, $w_{ij}$, this rule amounts to:

$$\Delta w_{ij} \propto \frac{\text{cof } w_{ij}}{\det \mathbf{W}} + x_j(1 - 2y_i) \tag{15}$$

where cof $w_{ij}$, the *cofactor* of $w_{ij}$, is $(-1)^{i+j}$ times the determinant of the matrix obtained by removing the $i$th row and the $j$th column from $\mathbf{W}$.

This rule is the same as the one for the single unit mapping, except that instead of $w = 0$ being an unstable point of the dynamics, now any degenerate weight matrix is unstable, since $\det \mathbf{W} = 0$ if $\mathbf{W}$ is degenerate. This fact enables different output units, $y_i$, to learn to represent different components in the input. When the weight vectors entering two output units become too similar, $\det \mathbf{W}$ becomes small and the natural dynamic of learning causes these weight vectors to diverge. This effect is mediated by the numerator, cof $w_{ij}$. When this cofactor becomes small, it indicates that there is a degeneracy in the weight matrix of the *rest* of the layer (ie: those weights not associated with input $x_j$ or output $y_i$). In this case, any degeneracy in $\mathbf{W}$ has less to do with the specific weight $w_{ij}$ that we are adjusting.

## 3   Finding independent components — blind separation

Maximising the information contained in a layer of units involves maximising the entropy of the individual units while minimising the mutual information (the *redundancy*) between them. Considering two units:

$$H(y_1, y_2) = H(y_1) + H(y_2) - I(y_1, y_2) \tag{16}$$

For $I(y_1, y_2)$ to be zero, $y_1$ and $y_2$ must be statistically independent of each other, so that $f_{y_1 y_2}(y_1, y_2) = f_{y_1}(y_1) f_{y_2}(y_2)$. Achieving such a representation is variously called factorial code learning, redundancy reduction (Barlow 1961, Atick 1992), or independent component analysis (ICA), and in the general case of continuously valued variables of arbitrary distributions, no learning algorithm has been shown to converge to such a representation.

Our method *will* converge to a minimum redundancy, factorial representation as long as the individual entropy terms in eq.16 do not override the redundancy term, making an $I(y_1, y_2) = 0$ solution sub-optimal. One way to ensure this cannot occur is if we have *a priori* knowledge of the general form of the pdfs of the independent components. Then we can tailor our choice of node-function to be optimal for transmitting information about these components. For example, unit distributions require piecewise linear node-functions for highest $H(y)$, while the more common gaussian forms require roughly sigmoidal curves. Once we have chosen our node-functions appropriately, we can be sure that an output node $y$ cannot have higher

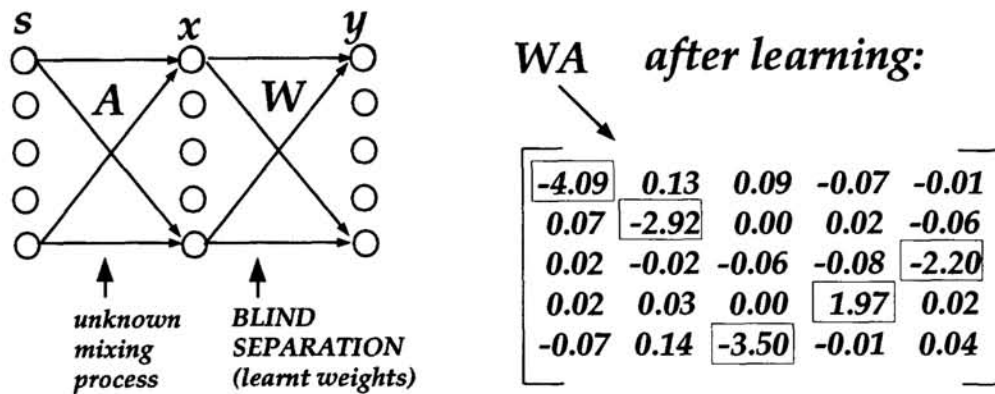

Figure 2: (a) In blind separation, sources, **s**, have been linearly scrambled by a matrix, **A**, to form the inputs to the network, **x**. We must recover the sources at our output **y**, by somehow inverting the mapping **A** with our weight matrix **W**. The problem: we know nothing about **A** or the sources. (b) A successful 'unscrambling' occurs when **WA** is a 'permutation' matrix. This one resulted from separating five speech signals with our algorithm.

entropy by representing some combination of independent components than by representing just one. When this condition is satisfied for all output units, the residual goal, of minimising the mutual information between the outputs, will dominate the learning. See [5] for further discussion of this.

With this caveat in mind, we turn to the problem of *blind separation*, (Jutten & Herault 1991), illustrated in Fig.2. A set of sources, $s_1, \ldots, s_N$, (different people speaking, music, noise etc) are presumed to be mixed approximately linearly so that all we receive is $N$ superpositions of them, $x_1, \ldots, x_N$, which are input to our single-layer information maximisation network. Providing the mixing matrix, **A**, is non-singular then the original sources can be recovered if our weight matrix, **W**, is such that **WA** is a 'permutation' matrix containing only one high value in each row and column.

Unfortunately we know nothing about the sources or the mixing matrix. However, if the sources are statistically independent and non-gaussian, then the information in the output nodes will be maximised when each output transmits one independent component only. This problem cannot be solved in general by linear decorrelation techniques such as those of (Barlow & Földiák 1989) since second-order statistics can only produce symmetrical decorrelation matrices.

We have tested the algorithm in eq.13 and eq.14 on digitally mixed speech signals, and it reliably converges to separate the individual sources. In one example, five separately-recorded speech signals from three individuals were sampled at 8kHz. Three-second segments from each were linearly mixed using a matrix of random values between 0.2 and 4. Each resulting mixture formed an incomprehensible babble. Time points were generated at random, and for each, the corresponding five mixed values were entered into the network, and weights were altered according to eq.13 and eq.14. After on the order of 500,000 points were presented, the network

had converged so that **WA** was the matrix shown in Fig.2b. As can be seen from the permutation structure of this matrix, on average, 95% of each output unit is dedicated to one source only, with each unit carrying a different source. Any residual interference from the four other sources was inaudible.

We have not yet performed any systematic studies on rates of convergence or existence of local minima. However the algorithm has converged to separate $N$ independent components in all our tests (for $2 \leq N \leq 10$). In contrast, we have not been able to obtain convergence of the H-J network on our data set for $N > 2$.

Finally, the kind of linear static mixing we have been using is not equivalent to what would be picked up by $N$ microphones positioned around a room. However, (Platt & Faggin 1992) in their work on the H-J net, discuss extensions for dealing with time-delays and non-static filtering, which may also be applicable to our methods.

## 4   Discussion

The problem of Independent Component Analysis (ICA) has become popular recently in the signal processing community, partly as a result of the success of the H-J network. The H-J network is identical to the linear decorrelation network of (Barlow & Földiák 1989) except for non-linearities in the anti-Hebb rule which normally performs only decorrelation. These non-linearities are chosen somewhat arbitrarily in the hope that their polynomial expansions will yield higher-order cross-cumulants useful in converging to independence (Comon et al, 1991). The H-J algorithm lacks an objective function, but these insights have led (Comon 1994) to propose minimising mutual information between outputs (see also Becker 1992). Since mutual information cannot be expressed as a simple function of the variables concerned, Comon expands it as a function of cumulants of increasing order.

In this paper, we have shown that mutual information, and through it, ICA, can be tackled directly (in the sense of eq.16) through a stochastic gradient approach. Sigmoidal units, being bounded, are limited in their 'channel capacity'. Weights transmitting information try, by following eq.13, to project their inputs to where they can make a lot of difference to the output, as measured by the log of the Jacobian of the transformation. In the process, each set of statistically 'dependent' information is channelled to the same output unit.

The non-linearity is crucial. If the network were just linear, the weight matrix would grow without bound since the learning rule would be:

$$\Delta \mathbf{W} \propto \left[ \mathbf{W}^T \right]^{-1} \tag{17}$$

This reflects the fact that the information in the outputs grows with their variance. The non-linearity also supplies the higher-order cross-moments necessary to maximise the infinite-order expansion of the information. For example, when $\mathbf{y} = \tanh(\mathbf{u})$, the learning rule has the form $\Delta \mathbf{W} \propto [\mathbf{W}^T]^{-1} - 2\mathbf{y}\mathbf{x}^T$, from which we can write that the weights stabilise (or $\langle \Delta \mathbf{W} \rangle = 0$) when $\mathbf{I} = 2\langle \tanh(\mathbf{u})\mathbf{u}^T \rangle$. Since tanh is an odd function, its series expansion is of the form $\tanh(u) = \sum_j b_j u^{2p+1}$, the $b_j$ being coefficients, and thus this convergence criterion amounts to the condition $\sum_{i,j} b_{ijp} \langle u_i^{2p+1} u_j \rangle = 0$ for all output unit pairs $i \neq j$, for $p = 0, 1, 2, 3 \ldots,$

and for the coefficients $b_{ijp}$ coming from the Taylor series expansion of the tanh function.

These and other issues are covered more completely in a forthcoming paper (Bell & Sejnowski 1995). Applications to blind deconvolution (removing the effects of unknown causal filtering) are also described, and the limitations of the approach are discussed.

## Acknowledgements

We are much indebted to Nici Schraudolph, who not only supplied the original idea in Fig.1 and shared his unpublished calculations [13], but also provided detailed criticism at every stage of the work. Much constructive advice also came from Paul Viola and Alexandre Pouget.

## Footnotes

[1]If $y = \tanh(wx + w_0)$ then $\Delta w \propto \frac{1}{w} - 2xy$

## References

[1] Atick J.J. 1992. Could information theory provide an ecological theory of sensory processing, *Network* 3, 213-251

[2] Barlow H.B. 1961. Possible principles underlying the transformation of sensory messages, in *Sensory Communication*, Rosenblith W.A. (ed), MIT press

[3] Barlow H.B. & Földiák P. 1989. Adaptation and decorrelation in the cortex, in Durbin R. et al (eds) *The Computing Neuron*, Addison-Wesley

[4] Becker S. 1992. An information-theoretic unsupervised learning algorithm for neural networks, *Ph.D. thesis*, Dept. of Comp. Sci., Univ. of Toronto

[5] Bell A.J. & Sejnowski T.J. 1995. An information-maximisation approach to blind separation and blind deconvolution, *Neural Computation*, in press

[6] Comon P., Jutten C. & Herault J. 1991. Blind separation of sources, part II: problems statement, *Signal processing*, 24, 11-21

[7] Comon P. 1994. Independent component analysis, a new concept? *Signal processing*, 36, 287-314

[8] Hopfield J.J. 1991. Olfactory computation and object perception, *Proc. Natl. Acad. Sci. USA*, vol. 88, pp.6462-6466

[9] Jutten C. & Herault J. 1991. Blind separation of sources, part I: an adaptive algorithm based on neuromimetic architecture, *Signal processing*, 24, 1-10

[10] Linsker R. 1992. Local synaptic learning rules suffice to maximise mutual information in a linear network, *Neural Computation*, 4, 691-702

[11] Papoulis A. 1984. *Probability, random variables and stochastic processes, 2nd edition*, McGraw-Hill, New York

[12] Platt J.C. & Faggin F. 1992. Networks for the separation of sources that are superimposed and delayed, in Moody J.E et al (eds) *Adv. Neur. Inf. Proc. Sys. 4*, Morgan-Kaufmann

[13] Schraudolph N.N., Hart W.E. & Belew R.K. 1992. Optimal information flow in sigmoidal neurons, *unpublished manuscript*